# The Belief in TAP

**Yoshiyuki Kabashima**
Dept. of Compt. Intl. & Syst. Sci.
Tokyo Institute of Technology
Yokohama 226, Japan

**David Saad**
Neural Computing Research Group
Aston University
Birmingham B4 7ET, UK

## Abstract

We show the similarity between belief propagation and TAP, for decoding corrupted messages encoded by Sourlas's method. The latter is a special case of the Gallager error-correcting code, where the code word comprises products of $K$ bits selected randomly from the original message. We examine the efficacy of solutions obtained by the two methods for various values of $K$ and show that solutions for $K \geq 3$ may be sensitive to the choice of initial conditions in the case of unbiased patterns. Good approximations are obtained generally for $K = 2$ and for biased patterns in the case of $K \geq 3$, especially when Nishimori's temperature is being used.

## 1 Introduction

Belief networks [1] are diagrammatic representations of joint probability distributions over a set of variables. This set is usually represented by the vertices of a graph, while arcs between vertices represent probabilistic dependencies between variables. Belief propagation provides a convenient mathematical tool for calculating iteratively joint probability distributions between variables and have been used in a variety of cases, most recently in the field of error correcting codes, for decoding corrupted messages [2] (for a review of graphical models and their use in the context of error-correcting codes see [3]).

Error-correcting codes provide a mechanism for retrieving the original message after corruption due to noise during transmission. Of a particular interest to the current paper is an error-correcting code presented by Sourlas [4] which is a special case of the Gallager codes [5]. The latter have been recently re-discovered by MacKay and Neal [2] and seem to have a significant practical potential.

In this paper we will examine the similarities between the belief propagation (BP) and TAP approaches, used to decode corrupted messaged encoded by Sourlas's method, and compare the solutions obtained by both approaches to the exact results obtained using the replica method [8]. The statistical mechanics approach will then

allow us to draw some conclusion on the efficacy of the TAP/BP approach in the context of error correcting codes.

The paper is arranged in the following manner: In section 2 we will introduce the encoding method and describe the decoding task. The Belief Propagation approach to the decoding process will be introduced in section 3 and will be compared to the TAP approach for diluted spin systems in section 4. Numerical solutions for various cases will be presented in section 5 and we will summarize our results and discuss their implications in section 6.

## 2 The decoding problem

In a general scenario, a message represented by an $N$ dimensional binary vector $\boldsymbol{\xi}$ is encoded by a vector $\boldsymbol{J}^0$ which is then transmitted through a noisy channel with some flipping probability $p$ per bit. The received message $\boldsymbol{J}$ is then decoded to retrieve the original message. Sourlas's code [4], is based on encoded message bits of the form $J^0_{i_1,i_2...i_K} = \xi_{i_1}\xi_{i_2}...\xi_{i_K}$, taking the product of different $K$ message sites for each code word bit.

In the statistical mechanics approach we will attempt to retrieve the original message by exploring the ground state of the following Hamiltonian which corresponds to the preferred state of the system in terms of 'energy'

$$\mathcal{H} = -\sum_{\langle i_1,...i_K\rangle} \mathcal{A}_{\langle i_1,...i_K\rangle}\, J_{\langle i_1,...i_K\rangle}\, S_{i_1}...S_{i_K} - F/\beta \sum_k S_k \;, \qquad (1)$$

where $\boldsymbol{S}$ is an $N$ dimensional binary vector of dynamical variables and $\mathcal{A}$ is a sparse tensor with $C$ unit elements per index (other elements are zero), which determines the components of $\boldsymbol{J}^0$. The last term on the right is required in the case of sparse (biased) messages and will require assigning a certain value to the additive field $F/\beta$, related to the prior belief in the Bayesian framework.

The statistical mechanical analysis can be easily linked to the Bayesian framework [4] in which one focuses on the posterior probability using Bayes theorem $\mathcal{P}(\boldsymbol{S}|\boldsymbol{J}) \sim \prod_\mu \mathcal{P}(J_\mu|\boldsymbol{S})\,\mathcal{P}_0(\boldsymbol{S})$ where $\mu$ runs over the message components and $\mathcal{P}_0(\boldsymbol{S})$ represents the prior. Knowing the posterior one can calculate the typical retrieved message elements and their alignment, which correspond to the Bayes-optimal decoding. The logarithms of the likelihood and prior terms are directly related to the first and second components of the Hamiltonian (Eq.1).

One should also note that $\mathcal{A}_{\langle i_1,...i_K\rangle} J_{\langle i_1,...i_K\rangle}$ represents a similar encoding scheme to that of Ref. [2] where a sparse matrix with $K$ non-zero elements per row multiplies the original message $\boldsymbol{\xi}$ and the resulting vector, modulo 2, is transmitted.

Sourlas analyzed this code in the cases of $K = 2$ and $K \to \infty$, where the ratio $C/K \to \infty$, by mapping them onto the SK [9] and Random Energy [10] models respectively. However, the ratio $R = K/C$ constitutes the code rate and the scenarios examined by Sourlas therefore correspond to the limited case of a vanishing code rate. The case of finite code rate, which we will consider here, has only recently been analyzed [8].

## 3 Decoding by belief propagation

As our goal, of calculating the posterior of the system $\mathcal{P}(\boldsymbol{S}|\boldsymbol{J})$ is rather difficult, we resort to the methods of BP, focusing on the calculation of conditional probabilities when some elements of the system are set to specific values or removed.

The approach adopted in this case, which is quite similar to the practical approach employed in the case of Gallager codes [2], assumes a two layer system corresponding to the elements of the corrupted message $\boldsymbol{J}$ and the dynamical variables $\boldsymbol{S}$ respectively, defining conditional probabilities which relate elements in the two layers:

$$q_{\mu l}^x = \mathcal{P}\left(S_l = x \mid \{J_{\nu \neq \mu}\}\right) \tag{2}$$

$$r_{\mu l}^x = \mathcal{P}\left(J_\mu | S_l = x, \{J_{\nu \neq \mu}\}\right) = \sum_{\{S_{k \neq l}\}} \mathcal{P}\left(J_\mu | S_l = x, \{S_{k \neq l}\}\right) \, \mathcal{P}\left(\{S_{k \neq l}\} | \{J_{\nu \neq \mu}\}\right) \, ,$$

where the index $\mu$ represents *an element* of the received vector message $\boldsymbol{J}$, constituted by a particular choice of indices $i_1, \ldots i_K$, which is connected to the corresponding index of $\boldsymbol{S}$ ($l$ in the first equation), i.e., for which the corresponding element $\mathcal{A}_{\langle i_1, \ldots i_K \rangle}$ is non-zero; the notation $\{S_{k \neq l}\}$ refers to all elements of $\boldsymbol{S}$, excluding the $l$-th element, which are connected to the corresponding index of $\boldsymbol{J}$ ($\mu$ in this case for the second equation); the index $x$ can take values of $\pm 1$. The conditional probabilities $q_{\mu l}^x$ and $r_{\mu l}^x$ will enable us, through recursive calculations to obtain an approximated expression to the posterior.

Employing Bayes rule and the assumption that the dependency of $S_l$ on an element $J_\nu$ is factorizable and vice versa: $\mathcal{P}\left(S_{l_1}, S_{l_2} \ldots S_{l_K} | \{J_{\nu \neq \mu}\}\right) = \prod_{k=1}^K \mathcal{P}\left(S_{l_k} | \{J_{\nu \neq \mu}\}\right)$ and $\mathcal{P}\left(\{J_{\nu \neq \mu}\} | S_l = x\right) = \prod_{\nu \neq \mu} \mathcal{P}\left(J_\nu | S_l = x, \{J_{\sigma \neq \nu}\}\right)$, one can rewrite a set of coupled equations for $q_{\mu l}^1$, $q_{\mu l}^{-1}$, $r_{\mu l}^1$ and $r_{\mu l}^{-1}$ of the form

$$q_{\mu l}^x = a_{\mu l} \, p_l^x \prod_{\nu \neq \mu} r_{\nu l}^x \quad \text{and} \quad r_{\mu l}^x = \sum_{\{S_{k \neq l}\}} \mathcal{P}\left(J_\mu | S_l = x, \{S_{k \neq l}\}\right) \prod_{k \neq l} q_{\mu k}^{S_k} \, , \tag{3}$$

where $a_{\mu l}$ is a normalizing factor such that $q_{\mu l}^1 + q_{\mu l}^{-1} = 1$ and $p_l^x = \mathcal{P}\left(S_l = x\right)$ are our prior beliefs in the value of the source bits $S_l$.

This set of equations can be solved iteratively [2] by updating a coupled set of difference equations for $\delta q_{\mu l} = q_{\mu l}^1 - q_{\mu l}^{-1}$ and $\delta r_{\mu l} = r_{\mu l}^1 - r_{\mu l}^{-1}$, derived for this specific model, making use of the fact that the variables $r_{\mu l}^x$, and sub-sequentially the variables $q_{\mu l}^x$, can be calculated by exploiting the relation $r_{\mu l}^{\pm 1} = (1 \pm \delta r_{\mu l})/2$ and Eq.(3). At each iteration we can also calculate the pseudo-posterior probabilities $q_l^x = a_l p_l^x \prod_\nu r_{\nu l}^x$, where $a_l$ are normalizing factors, to determine the current estimated value of $S_l$.

Two points that are worthwhile noting: Firstly, the iterative solution makes use of the normalization $r_{\mu l}^1 + r_{\mu l}^{-1} = 1$, which is *not* derived from the basic probability rules and makes implicit assumptions about the probabilities of obtaining $S_l = \pm 1$ for all elements $l$. Secondly, the iterative solution would have provided the true posterior probabilities $q_l^x$ if the graph connecting the message $\boldsymbol{J}$ and the encoded bits $\boldsymbol{S}$ would have been free of cycles, i.e., if the graph would have been a tree with no recurrent dependencies among the variables. The fact that the framework provides adequate practical solutions has only recently been explained [13].

## 4   Decoding by TAP

We will now show that for this particular problem it is possible to obtain a similar set of equations from the corresponding statistical mechanics framework based on Bethe approximation [11] or the TAP (Thouless-Anderson-Palmer) approach [12] to diluted systems [1] . In the statistical mechanics approach we assign a Boltzmann

weight to each set comprising an encoded message bit $J_\mu$ and a dynamical vector $\boldsymbol{S}$

$$w_B \left( J_\mu | \boldsymbol{S} \right) = e^{-\beta\ g\left(J_\mu | \boldsymbol{S}\right)} , \qquad (4)$$

such that the first term of the system's Hamiltonian (Eq.1) can be rewritten as $\sum_\mu g \left( J_\mu | \boldsymbol{S} \right)$, where the index $\mu$ runs over all non-zero sites in the multidimensional tensor $\mathcal{A}$. We will now employ two straightforward assumptions to write a set of coupled equations for the mean field $q_{\mu l}^{S_l} = \mathcal{P} \left( S_l | \{ J_{\nu \neq \mu} \} \right)$, which may be identified as the same variable as in the belief network framework (Eq.2), and the effective Boltzmann weight $w_{\text{eff}} (J_\mu | S_l, \{ J_{\nu \neq \mu} \})$:

1) we assume a mean field behavior for the dependence of the dynamical variables $\boldsymbol{S}$ on a certain realization of the message sites $\boldsymbol{J}$, i.e., the dependence is factorizable and may be replaced by a product of mean fields.

2) Boltzmann weights (effective) for site $S_l$ are factorizable with respect to $J_\mu$.

The resulting set of equations are of the form

$$w_{\text{eff}} \left( J_\mu \mid S_l, \{J_{\nu \neq \mu}\} \right) = \text{Tr}_{\{S_{k \neq l}\}}\ w_B \left( J_\mu \mid \boldsymbol{S} \right) \prod_{k \neq l} q_{\mu l}^{S_k}$$

$$q_{\mu l}^{S_l} = \tilde{a}_{\mu l}\ p_l^{S_l} \prod_{\nu \neq \mu} w_{\text{eff}} (J_\nu \mid S_l, \{J_{\sigma \neq \nu}\}) , \qquad (5)$$

where $\tilde{a}_{\mu l}$ is a normalization factor and $p_l^{S_l}$ is our prior knowledge of the source's bias. Replacing the effective Boltzmann weight by a normalized field, which may be identified as the variable $r_{\mu l}^{S_l}$ of Eq.(2), we obtain

$$r_{\mu l}^{S_l} = \mathcal{P} \left( S_l \mid J_\mu, \{J_{\nu \neq \mu}\} \right) = a_{\mu l}\ w_{\text{eff}} \left( J_\mu \mid S_l, \{J_{\nu \neq \mu}\} \right) , \qquad (6)$$

i.e., a set of equations equivalent to Eq.(3). The explicit expressions of the normalization coefficients, $a_{\mu l}$ and $\tilde{a}_{\mu l}$, are

$$a_{\mu l}^{-1} = \text{Tr}_{\{S\}}\ w_B \left( J_\mu | \boldsymbol{S} \right) \prod_{k \neq l} q_{\mu l}^{S_k} \quad \text{and} \quad \tilde{a}_{\mu l}^{-1} = \text{Tr}_{\{S_l\}}\ p_l^{S_l} \prod_{\nu \neq \mu} r_{\nu l}^{S_l} , \qquad (7)$$

The somewhat arbitrary use of the differences $\delta q_{\mu l} = \langle S_l^\mu \rangle_q$ and $\delta r_{\mu l} = \langle S_l^\mu \rangle_r$ in the BP approach becomes clear form the statistical mechanics description, where they represent the expectation values of the dynamical variables with respect to the fields. The statistical mechanics formulation also provides a partial answer to the successful use of the BP methods to loopy systems, as we consider a finite number of steps on an infinite lattice [14]. However, it does not provide an explanation in the case of small systems which should be examined using other methods.

The formulation so far has been general; however, in the case of Sourlas's code we can make use of the explicit expression for $g$ to derive the relation between $q_{\mu l}^{S_l}$, $r_{\mu l}^{S_l}$, $\delta q_{\mu l}$ and $\delta r_{\mu l}$ as well as an explicit expression for $w_B (J_\mu | \boldsymbol{S}, \beta)$

$$q_{\mu l}^{S_l} = \frac{1}{2} (1 + \delta q_{\mu l} S_l) \quad , \quad r_{\mu l}^{S_l} = \frac{1}{2} (1 + \delta r_{\mu l} S_l) \quad \text{and} \qquad (8)$$

$$w_B (J_\mu | \boldsymbol{S}, \beta) = \frac{1}{2} \cosh \beta J_\mu \left( 1 + \tanh \beta J_\mu \prod_{l \in \mathcal{L}(\mu)} S_l \right) , \qquad (9)$$

by the perturbation expansion of the mean field equations with respect to Onsager reaction fields since these fields are too large in diluted systems. Consequently, the resulting equations are different than those obtained for fully connected systems [12]. We termed our approach TAP, following the convention for the Bethe approximation when applied to disordered systems subject to mean field type random interactions.

where $\mathcal{L}(\mu)$ is the set of all sites of $\boldsymbol{S}$ connected to $J_\mu$, i.e., for which the corresponding element of the tensor $\mathcal{A}$ is non-zero. The explicit form of the equations for $\delta q_{\mu l}$ and $\delta r_{\mu l}$ becomes

$$\delta r_{\mu l} = \tanh \beta J_\mu \prod_{l \in \mathcal{L}(\mu)/l} \delta q_{\mu l} \quad \text{and} \quad \delta q_{\mu l} = \tanh \left( \sum_{\nu \in \mathcal{M}(l)/\mu} \tanh^{-1} \delta r_{\nu l} + F \right) , \quad (10)$$

where $\mathcal{M}(l)/\mu$ is the set of all indices of the tensor $\boldsymbol{J}$, excluding $\mu$, which are connected to the vector site $l$; the external field $F$ which previously appeared in the last term of Eq.(1) is directly related to our prior belief of the message bias

$$p_l^{S_l} = \frac{1}{2} \left( 1 + \tanh F S_l \right) . \quad (11)$$

We therefore showed that there is a direct relation between the equations derived from the BP approach and from TAP in this particular case. One should note that the TAP approach allows for the use of finite inverse-temperatures $\beta$ which is not naturally included in the BP approach.

## 5   Numerical solutions

To examine the efficacy of TAP/BP decoding we used the method for decoding corrupted messages encoded by the Sourlas scheme [4], for which we have previously obtained analytical solutions using the replica method [8]. We solved iteratively Eq.(10) for specific cases by making use of differences $\delta q_{\mu l}$ and $\delta r_{\mu l}$ to obtain the values of $q_{\mu l}^{\pm 1}$ and $r_{\mu l}^{\pm 1}$ and of the magnetization $M$.

Numerical solutions of 10 individual runs for each value of the flip rate $p$ starting from different initial conditions, obtained for the case $K = 2$ and $C = 4$, different biases ($f = p_l^1 = 0.1, 0.5$ - the probability of $+1$ bit in the original message $\boldsymbol{\xi}$) and temperatures ($T = 0.26, T_n$) are shown in Fig. 1a. For each run, 20000 bit code words $\boldsymbol{J}^0$ were generated from 10000 bit message $\boldsymbol{\xi}$ using a fixed random sparse tensor $\mathcal{A}$. The noise corrupted code word $\boldsymbol{J}$ was decoded to retrieve the original message $\boldsymbol{\xi}$. Initial conditions are set to $\delta r_{\mu l} = 0$ and $\delta q_{\mu l} = \tanh F$ reflecting the prior belief; whenever the TAP/BP approach was successful in predicting the theoretical values we observed convergence in most runs corresponding to the ferromagnetic phase while almost all runs at low temperatures did not converged to a stable solution above the critical flip-rate (although the magnetization $M$ did converge as one may expect). We obtain good agreement between the TAP/BP solutions and the theoretical values calculated using the methods of [8] (diamond symbols and dashed line respectively). The results for biased patterns at $T = 0.26$ presented in the form of mean values and standard deviation, show a sub-optimal improvement in performance as expected. Obtaining solutions under similar conditions but at Nishimori's temperature - $1/T_n = 1/2 \ln[(1 - p)/p]$ [7], we see that pattern sparsity is exploited optimally resulting in a magnetization $M \approx 0.8$ for high corruption rates, as $T_n$ simulates accurately the loss of information due to channel noise [6, 7]; results for unbiased patterns (not shown) are not affected significantly by the use of Nishimori's temperature.

The replica-based theoretical solutions [8] indicate a profoundly different behaviour for $K = 2$ in comparison to other $K$ values. We therefore obtained solutions for $K = 5$ under similar conditions (which are representative of results obtained in other cases of $K \neq 2$). The results presented in Fig. 1b, in terms of means and standard deviation of 10 individual runs per flip rate value $p$, are less encouraging as the iterative solutions are sensitive to the choice of initial conditions and tend to

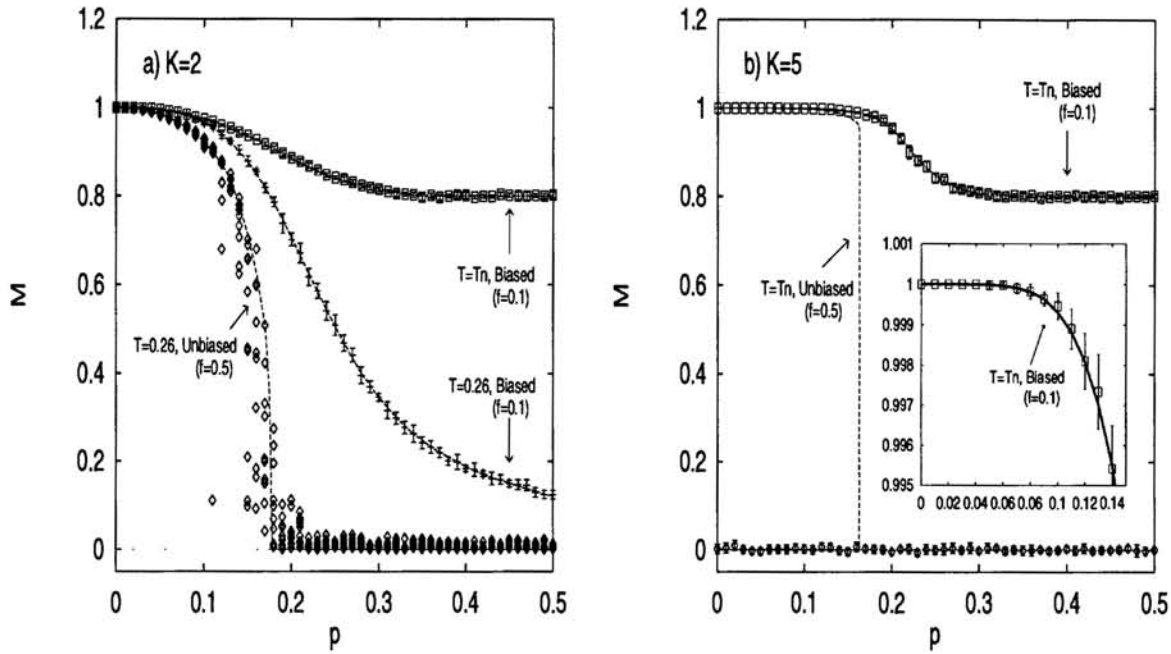

Figure 1: Numerical solutions for $M$ and different flip rate $p$. (a) For $K = 2$, different biases ($f = p_l^1 = 0.1, 0.5$) and temperatures ($T = 0.26, T_n$). Results for the unbiased patterns are shown as raw data (10 runs per flip rate value $p$ - diamond), while the theoretical solution is marked by the dashed line. Results for biased patterns are presented by their mean and standard deviation, showing a suboptimal performance as expected for $T = 0.26$ and an optimal one at Nishimori's temperature -$T_n$. The standard deviation is significantly smaller than the symbol size. Figure (b) shows results for the case $K = 5$ and $T = T_n$ in similar conditions to (a). Also here iterative solutions may generally drift away from the theoretical values where temperatures other than $T_n$ are employed (not shown); using Nishimori's temperature alleviates the problem only in the case of biased messages and the results are in close agreement with the theoretical solutions (inset - focusing on low $p$ values).

converge to sub-optimal values unless high sparsity and the appropriate choice of temperature ($T_n$) forces them to the correct values, showing then good agreement with the theoretical results (solid line, see inset). This phenomena is indicative of the fact that the ground state of the non-biased system is macroscopically degenerate with multiple equally good ground states.

We conclude that the TAP/BP approach may be highly useful in the case of biased patterns but may lead to errors for unbiased patterns and $K \geq 3$, and that the use of the appropriate temperature, i.e., Nishimori's temperature, enables one to obtain improved results, in agreement with results presented elsewhere [4, 6, 7].

# 6   Summary and discussion

We compared the use of BP to that of TAP for decoding corrupted messages encoded by Sourlas's method to discover that in this particular case the two methods provide a similar set of equations. We then solved the equations iteratively for specific cases and compared the results to those obtained by the replica method. The solutions indicate that the method is particularly useful in the case of biased messages and that using Nishimori's temperature is highly beneficial; solutions obtained using other temperature values may be sub-optimal. For non-sparse messages and $K \geq 3$ we may obtain erroneous solutions using these methods.

It would be desirable to explore whether the similarity in the equations derived using TAP and BP is restricted to this particular case or whether there is a more general link between the two methods. Another important question that remains open is the generality of our conclusions on the efficacy of these methods for decoding corrupted messages, as they are currently being applied in a variety of state-of-the-art coding schemes (e.g., [2, 3]). Understanding the limitations of these methods and the proper way to use them in general, especially in the context of error-correcting codes, may be highly beneficial to practitioners.

**Acknowledgment** This work was partially supported by the RFTF program of the JSPS (YK) and by EPSRC grant GR/L19232 (DS).

## Footnotes

[1] The terminology in the case of diluted systems is slightly vague. Unlike in the case of fully connected systems, self consistent equations of diluted systems cannot be derived

# References

[1] J. Pearl, *Probabilistic Reasoning in Intelligent Systems: Networks of Plausible Inference* (Morgan Kaufmann) 1988.

[2] D.J.C. MacKay and R.M. Neal, *Elect. Lett.*, **33**, 457 and preprint (1997).

[3] B.J. Frey, *Graphical Models for Machine Learning and Digital Communication* (MIT Press), 1998.

[4] N. Sourlas, *Nature*, **339**, 693 (1989) and *Europhys. Lett.*, **25**, 159 (1994).

[5] R.G. Gallager, *IRE Trans. Info. Theory*, **IT-8**, 21 (1962).

[6] P. Ruján, *Phys. Rev. Lett.*, **70**, 2968 (1993).

[7] H. Nishimori, *J. Phys. C*, **13**, 4071 (1980) and *J. Phys. Soc. of Japan*, **62**, 1169 (1993).

[8] Y. Kabashima and D. Saad, *Europhys. Lett.*, **45**, in press (1999).

[9] D. Sherrington and S. Kirkpatrick, *Phys. Rev. Lett.*, **35**, 1792 (1975).

[10] B. Derrida, *Phys. Rev. B*, **24**, 2613 (1981).

[11] H. Bethe, *Proc. R. Soc. A*, **151**, 540 (1935).

[12] D. Thouless, P.W. Anderson and R.G. Palmer, *Phil. Mag.*, **35**, 593 (1977).

[13] Y. Weiss, MIT preprint CBCL155 (1997).

[14] D. Sherrington and K.Y.M. Wong *J. Phys. A*, **20**, L785 (1987).